# A Nonlinear Predictive State Representation

**Matthew R. Rudary and Satinder Singh**
Computer Science and Engineering
University of Michigan
Ann Arbor, MI 48109
{mrudary,baveja}@umich.edu

## Abstract

Predictive state representations (PSRs) use predictions of a set of tests to represent the state of controlled dynamical systems. One reason why this representation is exciting as an alternative to partially observable Markov decision processes (POMDPs) is that PSR models of dynamical systems may be much more compact than POMDP models. Empirical work on PSRs to date has focused on *linear* PSRs, which have not allowed for compression relative to POMDPs. We introduce a new notion of tests which allows us to define a new type of PSR that is nonlinear in general and allows for exponential compression in some deterministic dynamical systems. These new tests, called *e-tests*, are related to the tests used by Rivest and Schapire [1] in their work with the diversity representation, but our PSR avoids some of the pitfalls of their representation—in particular, its potential to be exponentially larger than the equivalent POMDP.

## 1 Introduction

A predictive state representation, or PSR, captures the state of a controlled dynamical system not as a memory of past observations (as do history-window approaches), nor as a distribution over hidden states (as do partially observable Markov decision process or POMDP approaches), but as predictions for a set of tests that can be done on the system. A test is a sequence of action-observation pairs and the prediction for a test is the probability of the test-observations happening if the test-actions are executed. Littman et al. [2] showed that PSRs are as flexible a representation as POMDPs and are a more powerful representation than fixed-length history-window approaches. PSRs are potentially significant for two main reasons: 1) they are expressed entirely in terms of observable quantities and this may allow the development of methods for learning PSR models from observation data that behave and scale better than do existing methods for learning POMDP models from observation data, and 2) they may be much more compact than POMDP representations. It is the latter potential advantage that we focus on in this paper.

All PSRs studied to date have been linear, in the sense that the probability of any sequence of $k$ observations given a sequence of $k$ actions can be expressed as a linear function of the predictions of a core set of tests. We introduce a new type of test, the *e-test*, and present the first nonlinear PSR that can be applied to a general class of dynamical systems. In particular, in the first such result for PSRs we show that there exist controlled dynamical systems whose PSR representation is exponentially smaller than its POMDP representation.

To arrive at this result, we briefly review PSRs, introduce e-tests and an algorithm to generate a core set of e-tests given a POMDP, show that a representation built using e-tests is a PSR and that it can be exponentially smaller than the equivalent POMDP, and conclude with example problems and a look at future work in this area.

## 2  Models of Dynamical Systems

A model of a controlled dynamical system defines a probability distribution over sequences of observations one would get for any sequence of actions one could execute in the system. Equivalently, given any history of interaction with the dynamical system so far, a model defines the distribution over sequences of future observations for all sequences of future actions. The state of such a model must be a sufficient statistic of the observed history; that is, it must encode all the information conveyed by the history.

POMDPs [3, 4] and PSRs [2] both model controlled dynamical systems. In POMDPs, a belief state is used to encode historical information; in PSRs, probabilities of particular future outcomes are used. Here we describe both models and relate them to one another.

**POMDPs**   A POMDP model is defined by a tuple $\langle \mathcal{S}, \mathcal{A}, \mathcal{O}, T, O, b_0 \rangle$, where $\mathcal{S}$, $\mathcal{A}$, and $\mathcal{O}$ are, respectively, sets of (unobservable) hypothetical underlying-system states, actions that can be taken, and observations that may be issued by the system. $T$ is a set of matrices of dimension $|\mathcal{S}| \times |\mathcal{S}|$, one for each $a \in \mathcal{A}$, such that $T_{ij}^a$ is the probability that the next state is $j$ given that the current state is $i$ and action $a$ is taken. $O$ is a set of $|\mathcal{S}| \times |\mathcal{S}|$ diagonal matrices, one for each action-observation pair, such that $O_{ii}^{a;o}$ is the probability of observing $o$ after arriving in state $i$ by taking action $a$. Finally, $b_0$ is the initial belief state, a $|\mathcal{S}| \times 1$ vector whose $i^{th}$ element is the probability of the system starting in state $i$.

The belief state at history $h$ is $b(\mathcal{S}|h) = [prob(1|h) \ prob(2|h) \ \ldots \ prob(|\mathcal{S}||h)]$, where $prob(i|h)$ is the probability of the unobserved state being $i$ at history $h$. After taking an action $a$ in history $h$ and observing $o$, the belief state can be updated as follows:

$$b^T(\mathcal{S}|hao) = \frac{b^T(\mathcal{S}|h)T^a O^{a,o}}{b^T(\mathcal{S}|h)T^a O^{a,o}1_{|\mathcal{S}|}} \quad (1_{|\mathcal{S}|} \text{ is the } |\mathcal{S}| \times 1 \text{ vector consisting of all 1's})$$

**PSRs**   Littman et al. [2] (inspired by the work of Rivest and Schapire [1] and Jaeger [5]) introduced PSRs to represent the state of a controlled dynamical system using predictions of the outcomes of tests. They define a test $t$ as a sequence of actions and observations $t = a^1 o^1 a^2 o^2 \cdots a^k o^k$; we shall call this type of test a *sequence test*, or *s-test* in short. An s-test succeeds iff, when the sequence of actions $a^1 a^2 \cdots a^k$ is executed, the system issues the observation sequence $o^1 o^2 \cdots o^k$. The prediction $p(t|h)$ is the probability that the s-test $t$ succeeds from observed history $h$ (of length $n$ w.l.o.g.); that is

$$p(t|h) = prob(o_{n+1} = o^1, \ldots, o_{n+k} = o^k | h, a_{n+1} = a^1, \ldots, a_{n+k} = a^k) \quad (1)$$

where $a_i$ and $o_i$ denote the action taken and the observation, respectively, at time $i$. In the rest of this paper, we will abbreviate expressions like the right-hand side of Equation 1 by $prob(o^1 o^2 \cdots o^k | h a^1 a^2 \cdots a^k)$.

A set of s-tests $Q = \{q_1 q_2 \ldots q_{|Q|}\}$ is said to be a *core* set if it constitutes a PSR, i.e., if its vector of predictions, $p(Q|h) = [p(q_1|h) \ p(q_2|h) \ \ldots \ p(q_{|Q|}|h)]$, is a sufficient statistic for any history $h$. Equivalently, if $Q$ is a core set, then for any s-test $t$, there exists a function $f_t$ such that $p(t|h) = f_t(p(Q|h))$ for all $h$. The prediction vector $p(Q|h)$ in PSR models corresponds to belief state $b(\mathcal{S}|h)$ in POMDP models. The PSRs discussed by Littman et al. [2] are *linear* PSRs in the sense that for any s-test $t$, $f_t$ is a linear function of the predictions of the core s-tests; equivalently, the following equation

$$\forall \text{s-tests } t \ \exists \text{ a weight vector } w_t, s.t. \ p(t|h) = p^T(Q|h)w_t \quad (2)$$

defines what it means for $Q$ to constitute a linear PSR. Upon taking action $a$ in history $h$ and observing $o$, the prediction vector can be updated as follows:

$$p(q_i|hao) = \frac{p(aoq_i|h)}{p(ao|h)} = \frac{f_{aoq_i}(p(Q|h))}{f_{ao}(p(Q|h))} = \frac{p^T(Q|h)m_{aoq_i}}{p^T(Q|h)m_{ao}} \qquad (3)$$

where the final right-hand side is only valid for linear PSRs. Thus a linear PSR model is specified by $Q$ and by the weight vectors in the above equation $m_{aoq}$ for all $a \in \mathcal{A}, o \in \mathcal{O}, q \in Q \cup \phi$ (where $\phi$ is the null sequence). It is pertinent to ask what sort of dynamical systems can be modeled by a PSR and how many core tests are required to model a system. In fact, Littman et al. [2] answered these questions with the following result:

**Lemma 1** (Littman et al. [2]) *For any dynamical system that can be represented by a finite POMDP model, there exists a linear PSR model of size ($|Q|$) no more than the size ($|\mathcal{S}|$) of the POMDP model.*

Littman et al. prove this result by providing an algorithm for constructing a linear PSR model from a POMDP model. The algorithm they present depends on the insight that s-tests are differentiated by their *outcome vectors*. An outcome vector $u(t)$ for an s-test $t = a^1o^1a^2o^2 \ldots a^ko^k$ is a $|S| \times 1$ vector; the $i^{th}$ component of the vector is the probability of $t$ succeeding given that the system is in the hidden state $i$, i.e., $u(t) = T^{a^1}O^{a^1o^1}T^{a^2}O^{a^2o^2} \ldots T^{a^n}O^{a^ko^k}1_{|\mathcal{S}|}$. Consider the matrix $U$ whose rows correspond to the states in $\mathcal{S}$ and whose columns are the outcome vectors for all possible s-tests. Let $Q$ denote the set of s-tests associated with the maximal set of linearly independent columns of $U$; clearly $|Q| \leq |\mathcal{S}|$. Littman et al. showed that $Q$ is a core set for a linear PSR model by the following logic. Let $U(Q)$ denote the submatrix consisting of the columns of $U$ corresponding to the s-tests $\in Q$. Clearly, for any s-test $t$, $u(t) = U(Q)w_t$ for some vector of weights $w_t$. Therefore, $p(t|h) = b^T(\mathcal{S}|h)u(t) = b^T(\mathcal{S}|h)U(Q)w_t = p(Q|h)w_t$ which is exactly the requirement for a linear PSR (cf. Equation 2).

We will reuse the concept of linear independence of outcome vectors with a new type of test to derive a PSR that is nonlinear in general. This is the first nonlinear PSR that can be used to represent a general class of problems. In addition, this type of PSR in some cases requires a number of core tests that is exponentially smaller than the number of states in the minimal POMDP or the number of core tests in the linear PSR.

## 3   A new notion of tests

In order to formulate a PSR that requires fewer core tests, we look to a new kind of test—the end test, or *e-test* in short. An e-test is defined by a sequence of actions and a single ending observation. An e-test $e = a^1a^2 \cdots a^ko^k$ succeeds if, after the sequence of actions $a^1a^2 \cdots a^k$ is executed, $o^k$ is observed. This type of test is inspired by Rivest and Schapire's [1] notion of tests in their work on modeling *deterministic* dynamical systems.

### 3.1   PSRs with e-tests

Just as Littman et al. considered the problem of constructing s-test-based PSRs from POMDP models, here we consider how to construct a e-test-based PSR, or EPSR, from a POMDP model and will derive properties of EPSRs from the resulting construction.

The $|\mathcal{S}| \times 1$ outcome vector for an e-test $e = a^1a^2 \ldots a^ko^k$ is

$$v(e) = T^{a^1}T^{a^2} \ldots T^{a^k}O^{a^ko^k}1_{|\mathcal{S}|}. \qquad (4)$$

Note that we are using $v$'s to denote outcome vectors for e-tests and $u$'s to denote outcome vectors for s-tests. Consider the matrix $V$ whose rows correspond to $\mathcal{S}$ whose columns are

```
done ← false; i ← 0; L ← {}
do until  done
    done ← true
    N ← generate all one-action extensions of length-i tests in L
    for  each t ∈ N
        if v(t) is linearly independent of V(L) then
            L ← L ∪ {t}; done ← false
    end for
    i ← i + 1
end do
Q_V ← L
```

Figure 1: Our search algorithm to find a set of core e-tests given the outcome vectors.

the outcome vectors for all possible e-tests. Let $Q_V$ denote the set of e-tests associated with a maximal set of linearly independent columns of matrix $V$; clearly $|Q_V| \leq |\mathcal{S}|$. Note that $Q_V$ is not uniquely defined; there are many such sets. The hope is that the set $Q_V$ is a core set for an EPSR model of the dynamical system represented by the POMDP model. But before we consider this hope, let us consider how we would find $Q_V$ given a POMDP model.

We can compute the outcome vector for any e-test from the POMDP parameters using Equation 4. So we could compute the columns of $V$ one by one and check to see how many linearly independent columns we find. If we ever find $|\mathcal{S}|$ linearly independent columns, we know we can stop, because we will not find any more. However, if $|Q_V| < |\mathcal{S}|$, then how would we know when to stop? In Figure 1, we present a search algorithm that finds a set $Q_V$ in polynomial time. Our algorithm is adapted from Littman et al.'s algorithm for finding core s-tests. The algorithm starts with all e-tests of length one and maintains a set $L$ of currently known linearly independent e-tests. At each iteration it searches for new linearly independent e-tests among all one-action extensions of the e-tests it added to $L$ at the last iteration and stops when an iteration does not add to the set $L$.

**Lemma 2**  *The search algorithm of Figure 1 computes the set $Q_V$ in time polynomial in the size of the POMDP.*

**Proof**    Computing the outcome vector for an e-test using Equation 4 is polynomial in the number of states and the length of the e-test. There cannot be more than $|\mathcal{S}|$ e-tests in the set $L$ maintained by the search algorithm algorithm and only one-action extensions of the e-tests in $L \cup \mathcal{O}$ are ever considered. Each extension length considered must add an e-test else the algorithm stops, and so the maximal length of each e-test in $Q_V$ is upper bounded by the number of states. Therefore, our algorithm returns $Q_V$ in polynomial time.    □

Note that this algorithm is only practical if the outcome vectors have been found; this only makes sense if the POMDP model is already known, as outcome vectors map POMDP states to outcomes. We will address learning these models from observations in future work [6]. Next we show that the prediction of any e-test can be computed linearly from the prediction vector for the e-tests in $Q_V$.

**Lemma 3**  *For any history $h$ and any e-test $e$, the prediction $p(e|h)$ is some linear function of prediction vector $p(Q_V|h)$, i.e., $p(e|h) = p(Q_V|h)w_e$ for some weight vector $w_e$.*

**Proof**    Let $V(Q_V)$ be the submatrix of $V$ containing the columns corresponding to $Q_V$. By the definition of $Q_V$, for any e-test $e$, $v(e) = V(Q_V)w_e$, for some weight vector $w_e$. Furthermore, for any history $h$, $p(e|h) = b(\mathcal{S}|h)v(e) = b(\mathcal{S}|h)V(Q_V)w_e = p(Q_V|h)w_e$.
    □

Note that Lemma 3 does not imply that $Q_V$ constitutes a PSR, let alone a linear PSR, for the definition of a PSR requires that the prediction of all *s-tests* be computable from the core test predictions. Next we turn to the crucial question: does $Q_V$ constitute a PSR?

**Theorem 1** *If $V(Q_V)$, defined as above with respect to some POMDP model of a dynamical system, is a square matrix, i.e., the number of e-tests in $Q_V$ is the number of states $|\mathcal{S}|$ (in that POMDP model), then $Q_V$ constitutes a **linear** EPSR for that dynamical system.*

**Proof** For any history $h$, $p^T(Q_V|h) = b^T(\mathcal{S}|h)V(Q_V)$. If $V(Q_V)$ is square then it is invertible because by construction it has full rank, and hence for any history $h$, $b^T(\mathcal{S}|h) = p^T(Q_V|h)V^{-1}(Q_V)$. For any s-test $t = a^1 o^1 a^2 o^2 \cdots a^k o^k$,

$$p^T(t|h) = b^T(\mathcal{S}|h)T^{a^1}O^{a^1,o^1}T^{a^2}O^{a^2,o^2}\cdots T^{a^k}O^{a^k,o^k}1_\mathcal{S} \text{ (by first-principles definition)}$$
$$= p^T(Q_V|h)V^{-1}(Q_V)T^{a^1}O^{a^1,o^1}T^{a^2}O^{a^2,o^2}\cdots T^{a^k}O^{a^k,o^k}1_\mathcal{S} = p^T(Q_V|h)w_t$$

for some weight vector $w_t$. Thus, $Q_V$ constitutes a linear EPSR as per the definition in Equation 2. $\square$

We note that the product $T^{a^1}O^{a^1,o^1}\cdots T^{a^k}O^{a^k,o^k}1_\mathcal{S}$ appears often in association with an s-test $t = a^1 o^1 \cdots a^k o^k$, and abbreviate the product $z(t)$. We similarly define $z(e) = T^{a^1}T a^2 \cdots T^{a^k}O^{a^k,o^k}1_\mathcal{S}$ for the e-test $e = a^1 a^2 \cdots a^k o^k$.

Staying with the linear EPSR case for now, we can define an *update function* for $p(Q_V|h)$ as follows: (remembering that $V(Q_V)$ is invertible for this case)

$$p(e_i|hao) = \frac{p(aoe_i|h)}{p(ao|h)} = \frac{b(\mathcal{S}|h)T^a O^{a,o}z(e_i)}{p(Q|h)m_{ao}} = \frac{p(Q_V|h)V^{-1}(Q_V)z(aoe_i)}{p(Q_V|h)m_{ao}} = \frac{p(Q_V|h)m_{aoe_i}}{p(Q_V|h)m_{ao}}$$
$$(5)$$

where we used the fact that the test $ao$ in the denominator is an e-test. (The form of the linear EPSR update equation is identical to the form of the update in linear PSRs with s-tests given in Equation 3). Thus, a linear EPSR model is defined by $Q_V$ and the set of weight vectors, $m_{aoe}$ for all $a \in \mathcal{A}, o \in \mathcal{O}, e \in \{Q_V \cup \phi\}$, used in Equation 5.

Now, let us turn to the case when the number of e-tests in $Q_V$ is less than $|S|$, i.e., when $V(Q_V)$ is not a square matrix.

**Lemma 4** *In general, if the number of e-tests in $Q_V$ is less than $|S|$, then $Q_V$ is **not** guaranteed to be a linear EPSR.*

**Proof** (Sketch) To prove this lemma, we must only find an example of a dynamical system that is an EPSR but not a linear EPSR. In Section 4.3 we present a $k$-bit register as an example of such a problem. We show in that section that the state space size is $2^k$ and the number of s-tests in the core set of a linear s-test-based PSR model is also $2^k$, but the number of e-tests in $Q_V$ is only $k + 1$. Note that this means that the rank of the $U$ matrix for s-tests is $2^k$ while the rank of the $V$ matrix for e-tests is $k + 1$. This must mean that $Q_V$ is not a linear EPSR. We skip these details for lack of space. $\square$

Lemma 4 leaves open the possibility that if $|Q_V| < |\mathcal{S}|$ then $Q_V$ constitutes a nonlinear EPSR. We were unable, thus far, to evaluate this possibility for general POMDPs but we did obtain an interesting and positive answer, presented in the next section, for the class of deterministic POMDPs.

# 4 A Nonlinear PSR for Deterministic Dynamical Systems

In deterministic dynamical systems, the predictions of both e-tests and s-tests are binary and it is this basic fact that allows us to prove the following result.

**Theorem 2** *For deterministic dynamical systems the set of e-tests $Q_V$ is always an EPSR and in general it is a nonlinear EPSR.*

**Proof** For an arbitrary s-test $t = a^1 o^1 a^2 o^2 \cdots a^k o^k$, and some arbitrary history $h$ that is realizable (i.e., $p(h) = 1$), and for some vectors $w_{a^1 o^1}$, $w_{a^1 o^1 a^2 o^2}$, ..., $w_{a^1 o^1 a^2 o^2 \cdots a^k o^k}$ of length $|Q_V|$, we have

$$
\begin{aligned}
prob(o^1 o^2 \cdots o^k | h a^1 a^2 \cdots a^k) & = \\
& = prob(o^1 | h a^1) prob(o^2 | h a^1 o^1 a^2) \cdots prob(o^k | h a^1 o^1 a^2 o^2 \cdots a^{k-1} o^{k-1} a^k) \\
& = prob(o^1 | h a^1) prob(o^2 | h a^1 a^2) \cdots prob(o^k | h a^1 a^2 \cdots a^k) \\
& = (p^T(Q_V | h) w_{a^1 o^1})(p^T(Q_V | h) w_{a^1 o^1 a^2 o^2}) \cdots (p^T(Q_V | h) w_{a^1 o^1 \cdots a^k o^k}) \\
& = f_t(p(Q_V | h))
\end{aligned}
$$

In going from the second line to the third, we eliminate observations from the conditions by noting that in a deterministic system, the observation emitted depends only on the sequence of actions executed. In going from the third line to the fourth, we use the result of Lemma 3 that regardless of the size of $Q_V$, the predictions for all e-tests for any history $h$ are linear functions of $p(Q_V | h)$. This shows that for deterministic dynamical systems, $Q_V$, regardless of its size, constitutes an EPSR. □

**Update Function:** Since predictions are binary in deterministic EPSRs, $p(ao|h)$ must be 1 if $o$ is observed after taking action $a$ in history $h$:

$$
p(e_i | hao) = p(aoe_i | h) / p(ao | h) = p(ae_i | h) = p(Q_V | h) m_{ae_i}
$$

where the second equality from the left comes about because $p(ao|h) = 1$ and, because $o$ must be observed when $a$ is executed, $p(aoe_i | h) = p(ae_i | h)$, and the last equality used the fact that $ae_i$ is just some other e-test and so from Lemma 3 must be a linear function of $p(Q_V | h)$. It is rather interesting that even though the EPSR formed through $Q_V$ is nonlinear (as seen in Theorem 2), the update function is in fact linear.

### 4.1 Diversity and e-tests

Rivest and Schapire's [1] diversity representation, the inspiration for e-tests, applies only to deterministic systems and can be explained using the binary outcome matrix $V$ defined at the beginning of Section 3.1. Diversity also uses the predictions of a set of e-tests as its representation of state; it uses as many e-tests as there are distinct columns in the matrix $V$. Clearly, at most there can be $2^{|\mathcal{S}|}$ distinct columns and they show that there have to be at least $log_2(|\mathcal{S}|)$ distinct columns and that these bounds are tight. Thus the size of the diversity representation can be exponentially smaller or exponentially bigger than the size of a POMDP representation. While we use the same notion of tests as the diversity representation, our use of linear independence of outcome vectors instead of equivalence classes based on equality of outcome vectors allows us to use e-tests on stochastic systems.

Next we show through an example that EPSR models in deterministic dynamic systems can lead to exponential compression over POMDP models in some cases while avoiding the exponential blowup possible in Rivest and Schapire's [1] diversity representation.

### 4.2 EPSRs can be Exponentially Smaller than Diversity

This first example shows a case in which the size of the EPSR representation is exponentially smaller than the size of the diversity representation. The hit register (see Figure 2a) is a $k$-bit register (these are the *value bits*) with an additional special *hit bit*. There are $2^k + 1$ states in the POMDP describing this system—one state in which the hit bit is 1 and $2^k$ states in which the hit bit is 0 and the value bits take on different combinations of

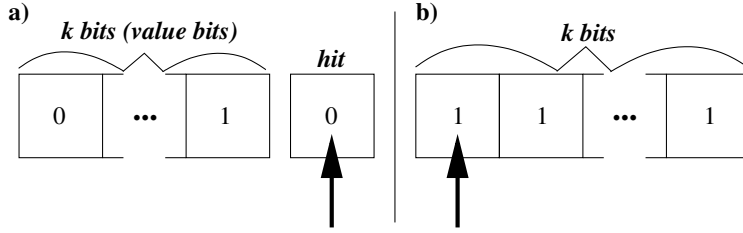

Figure 2: The two example systems. a) The $k$-bit hit register. There are $k$ value bits and the special hit bit. The value of the hit bit determines the observation and $k + 2$ actions alter the value of the bits; this is fully specified in Section 4.2. b) The $k$-bit rotate register. The value of the leftmost bit is observed; this bit can be flipped, and the register can be rotated either to the right or to the left. This is described in greater detail in Section 4.3.

values. There are $k + 2$ actions: a flip action $F_i$ for each value bit $i$ that inverts bit $i$ if the hit bit is not set, a set action $S_h$ that sets the hit bit if all the value bits are 0, and a clear action $C_h$ that clears the hit bit. There are two observations: $O_h$ if the hit bit is set and $O_m$ otherwise. Rivest and Schapire [1] present a similar problem (their version has no $C_h$ action). The diversity representation requires $O(2^{2^k})$ equivalence classes and thus tests, whereas an EPSR has only $2^k + 1$ core e-tests (see Table 1 for the core e-tests and update function when $k = 2$).

Table 1: Core e-tests and update functions for the 2-bit hit register problem.

| test | update function for action | | | |
|---|---|---|---|---|
| | $F_1$ | $F_2$ | $S_h$ | $C_h$ |
| $F_1 O_h$ | $p(F_1 O_h)$ | $p(F_1 O_h)$ | $p(S_h O_h)$ | 0 |
| $S_h O_h$ | $p(F_1 S_h O_h)$ | $p(F_2 S_h O_h)$ | $p(S_h O_h)$ | $p(S_h O_h)$ |
| $F_1 S_h O_h$ | $p(S_h O_h)$ | $p(F_2 F_1 S_h O_h)$ | $p(S_h O_h) - p(F_1 O_h) + p(F_1 S_h O_h)$ | $p(F_1 S_h O_h) - p(F_1 O_h)$ |
| $F_2 S_h O_h$ | $p(F_2 F_1 S_h O_h)$ | $p(S_h O_h)$ | $p(S_h O_h) - p(F_1 O_h) + p(F_2 S_h O_h)$ | $p(F_2 S_h O_h) - p(F_1 O_h)$ |
| $F_2 F_1 S_h O_h$ | $p(F_2 S_h O_h)$ | $p(F_1 S_h O_h)$ | $p(S_h O_h) - p(F_1 O_h) + p(F_2 F_1 S_h O_h)$ | $p(F_2 F_1 S_h O_h) - p(F_1 O_h)$ |

**Lemma 5** *For deterministic dynamical systems, the size of the EPSR representation is always upper-bounded by the* minimum *of the size of the diversity representation and the size of the POMDP representation.*

**Proof**  The size of the EPSR representation, $|Q_V|$, is upper-bounded by $|\mathcal{S}|$ by construction of $Q_V$. The number of e-tests used by diversity representation is the number of distinct columns in the binary $V$ matrix of Section 3.1, while the number of e-tests used by the EPSR representation is the number of linearly independent columns in $V$. Clearly the latter is upper-bounded by the former. As a quick example, consider the case of 2-bit binary vectors: There are 4 distinct vectors but only 2 linearly independent ones. $\square$

### 4.3  EPSRs can be Exponentially Smaller than POMDPs and the Original PSRs

This second example shows a case in which the EPSR representation uses exponentially fewer tests than the number of states in the POMDP representation as well as the original linear PSR representation. The rotate register illustrated in Figure 2b is a $k$-bit shift-register.

Table 2: Core e-tests and update function for the 4 bit rotate register problem.

| test | update function for action | | |
|---|---|---|---|
| | $R$ | $L$ | $F$ |
| $FO_1$ | $p(FO_1) + p(FFO_1) - p(RO_1)$ | $p(FO_1) + p(FFO_1) - p(LO_1)$ | $p(FFO_1)$ |
| $RO_1$ | $p(RRO_1)$ | $p(FFO_1)$ | $p(RO_1)$ |
| $LO_1$ | $p(FFO_1)$ | $p(RRO_1)$ | $p(LO_1)$ |
| $FFO_1$ | $p(RO_1)$ | $p(LO_1)$ | $p(FO_1)$ |
| $RRO_1$ | $p(LO_1)$ | $p(RO_1)$ | $p(RRO_1)$ |

There are two observations: $O_1$ is observed if the leftmost bit is $1$ and $O_0$ is observed when the leftmost bit is $0$. The three actions are $R$, which shifts the register one place to the right with wraparound, $L$, which does the opposite, and $F$, which flips the leftmost bit. This problem is also presented by Rivest and Schapire as an example of a system whose diversity is exponentially smaller than the number of states in the minimal POMDP, which is $2^k$. This is also the number of core s-tests in the equivalent linear PSR (we computed these $2^k$ s-tests but do not report them here). The diversity is $2k$. However, the EPSR that models this system has only $k + 1$ core e-tests. The tests and update function for the 4-bit rotate register are shown in Table 2.

## 5    Conclusions and Future Work

In this paper we have used a new type of test, the e-test, to specify a nonlinear PSR for deterministic controlled dynamical systems. This is the first nonlinear PSR for any general class of systems. We proved that in some deterministic systems our new PSR models are exponentially smaller than both the original PSR models as well as POMDP models. Similarly, compared to the size of Rivest & Schapire's diversity representation (the inspiration for the notion of e-tests) we proved that our PSR models are never bigger but sometimes exponentially smaller. This work has primarily been an attempt to understand the representational implications of using e-tests; as future work, we will explore the computational implications of switching to e-tests.

### Acknowledgments

Matt Rudary and Satinder Singh were supported by a grant from the Intel Research Council.

### References

[1] Ronald L. Rivest and Robert E. Schapire. Diversity-based inference of finite automata. *Journal of the ACM*, 41(3):555–589, May 1994.

[2] Michael L. Littman, Richard S. Sutton, and Satinder Singh. Predictive representations of state. In *Advances In Neural Information Processing Systems 14*, 2001.

[3] William S. Lovejoy. A survey of algorithmic methods for partially observed markov decision processes. *Annals of Operations Research*, 28(1):47–65, 1991.

[4] Michael L. Littman. *Algorithms for Sequential Decision Making*. PhD thesis, Brown University, 1996.

[5] Herbert Jaeger. Observable operator models for discrete stochastic time series. *Neural Computation*, 12(6):1371–1398, 2000.

[6] Satinder Singh, Michael L. Littman, Nicholas E. Jong, David Pardoe, and Peter Stone. Learning predictive state representations. In *The Twentieth International Conference on Machine Learning (ICML-2003)*, 2003. To appear.
